# Generalizable Singular Value Decomposition for Ill-posed Datasets

**Ulrik Kjems**     **Lars K. Hansen**
Department of Mathematical Modelling
Technical University of Denmark
DK-2800 Kgs. Lyngby, Denmark
*uk,lkhansen@imm.dtu.dk*

**Stephen C. Strother**
PET Imaging Service
VA medical center
Minneapolis
*steve@pet.med.va.gov*

## Abstract

We demonstrate that statistical analysis of ill-posed data sets is subject to a bias, which can be observed when projecting independent test set examples onto a basis defined by the training examples. Because the training examples in an ill-posed data set do not fully span the signal space the observed training set variances in each basis vector will be too high compared to the average variance of the test set projections onto the same basis vectors. On basis of this understanding we introduce the Generalizable Singular Value Decomposition (GenSVD) as a means to reduce this bias by re-estimation of the singular values obtained in a conventional Singular Value Decomposition, allowing for a generalization performance increase of a subsequent statistical model. We demonstrate that the algorithm succesfully corrects bias in a data set from a functional PET activation study of the human brain.

## 1   Ill-posed Data Sets

An ill-posed data set has more dimensions in each example than there are examples. Such data sets occur in many fields of research typically in connection with image measurements. The associated statistical problem is that of extracting structure from the observed high-dimensional vectors in the presence of noise. The statistical analysis can be done either supervised (i.e. modelling with target values: classification, regresssion) or unsupervised (modelling with no target values: clustering, PCA, ICA). In both types of analysis the ill-posedness may lead to immediate problems if one tries to apply conventional statistical methods of analysis, for example the empirical covariance matrix is prohibitively large and will be rank-deficient.

A common approach is to use Singular Value Decomposition (SVD) or the analogue Principal Component Analysis (PCA) to reduce the dimensionality of the data. Let the $N$ observed $i$-dimensional samples $\boldsymbol{x}_j$, $j = 1...N$, collected in the data matrix $\boldsymbol{X} = [\boldsymbol{x}_1 ... \boldsymbol{x}_N]$ of size $I \times N$, $I > N$. The SVD-theorem states that such a matrix can be decomposed as

$$\boldsymbol{X} = \boldsymbol{U}\boldsymbol{\Lambda}\boldsymbol{V}^\mathsf{T}, \tag{1}$$

where $U$ is a matrix of the same size as $X$ with orthogonal basis vectors spanning the space of $X$, so that $U^\mathsf{T}U = I_{N \times N}$. The square matrix $\Lambda$ contains the singular values in the diagonal, $\Lambda = \text{diag}(\lambda_1, ..., \lambda_N)$, which are ordered and positive $\lambda_1 \geq \lambda_2 \geq ... \geq \lambda_N \geq 0$, and $V$ is $N \times N$ and orthogonal $V^\mathsf{T}V = I_N$. If there is a mean value significantly different from zero it may at times be advantageous to perform the above analysis on mean-subtracted data, i.e. $X - \bar{X} = U\Lambda V^\mathsf{T}$ where columns of $\bar{X}$ all contain the mean vector $\bar{x} = \sum_j x_j / N$.

Each observation $x_j$ can be expressed in coordinates in the basis defined by the vectors of $U$ with no loss of information[Lautrup et al., 1995]. A change of basis is obtained by $q_j = U^\mathsf{T}x_j$ as the orthogonal basis rotation

$$Q = [q_1 \, ... \, q_N] = U^\mathsf{T}X = U^\mathsf{T}U\Lambda V^\mathsf{T} = \Lambda V^\mathsf{T}. \tag{2}$$

Since $Q$ is only $N \times N$ and $N \ll I$, $Q$ is a compact representation of the data. Having now $N$ examples of $N$ dimension we have reduced the problem to a marginally ill-posed one. To further reduce the dimensionality, it is common to retain only a subset of the coordinates, e.g. the top $P$ coordinates $(P < N)$ and the supervised or unsupervised model can be formed in this smaller but now well-posed space.

So far we have considered the procedure for modelling from a training set. Our hope is that the statistical description generalizes well to new examples proving that is is a good description of the generating process. The model should, in other words, be able to perform well on a new example, $x^*$, and in the above framework this would mean the predictions based on $q^* = U^\mathsf{T}x^*$ should generalize well. We will show in the following, that in general, *the distribution of the test set projection $q^*$ is quite different from the statistics of the projections of the training examples $q_j$*. It has been noted in previous work [Hansen and Larsen, 1996, Roweis, 1998, Hansen et al., 1999] that PCA/SVD of ill-posed data does not by itself represent a probabilistic model where we can assign a likelihood to a new test data point, and procedures have been proposed which make this possible. In [Bishop, 1999] PCA has been considered in a Bayesian framework, but does not address the significant bias of the variance in training set projections in ill-posed data sets. In [Jackson, 1991] an asymptotic expression is given for the bias of eigen-values in a sample covariance matrix, but this expression is valid only in the well-posed case and is not applicable for ill-posed data.

## 1.1 Example

Let the signal source be $I$-dimensional multivariate Gaussian distribution $\mathcal{N}(0, \Sigma)$ with a covariance matrix where the first $K$ eigen-values equal $\sigma^2$ and the last $I - K$ are zero, so that the covariance matrix has the decomposition

$$\Sigma = \sigma^2 Y D Y^\mathsf{T}, \qquad D = \text{diag}(1, ..., 1, 0, ..., 0), \qquad Y^\mathsf{T}Y = I \tag{3}$$

Our $N$ samples of the distribution are collected in the matrix $X = [x_{ij}]$ with the SVD

$$X = U\Lambda V^\mathsf{T}, \qquad \Lambda = \text{diag}(\lambda_1, ..., \lambda_N) \tag{4}$$

and the representation of the $N$ examples in the $N$ basis vector coordinates defined by $U$ is $Q = [q_{ij}] = U^\mathsf{T}X = \Lambda V^\mathsf{T}$. The total variance per training example is

$$\frac{1}{N}\sum_{i,j} x_{ij}^2 \;=\; \frac{1}{N}\text{Tr}(X^\mathsf{T}X) = \frac{1}{N}\text{Tr}(V\Lambda U^\mathsf{T}U\Lambda V^\mathsf{T}) = \frac{1}{N}\text{Tr}(V\Lambda^2 V^\mathsf{T})$$

$$=\; \frac{1}{N}\text{Tr}(VV^\mathsf{T}\Lambda^2) = \frac{1}{N}\text{Tr}(\Lambda^2) = \frac{1}{N}\sum_i \lambda_i^2 \tag{5}$$

Note that this variance is the same in the $U$-basis coordinates:

$$\frac{1}{N}\sum_{i,j} q_{ij}^2 \;=\; \frac{1}{N}\mathrm{Tr}(Q^\mathsf{T} Q) = \frac{1}{N}\mathrm{Tr}(V\Lambda^2 V^\mathsf{T}) = \frac{1}{N}\sum_i \lambda_i^2 \qquad (6)$$

We can derive the expected value of this variance:

$$\left\langle \frac{1}{N}\sum_{i,j} x_{j,i}^2 \right\rangle \;=\; \left\langle \sum_i x_{1,i}^2 \right\rangle = \langle x_1^\mathsf{T} x_1 \rangle = \mathrm{Tr}\Sigma = \sigma^2 K \qquad (7)$$

Now, consider a test example $x^* \sim \mathcal{N}(0,\Sigma)$ with the projection $q^* = U^\mathsf{T} x^*$ which will have the average total variance

$$\sum_i q_i^{*2} \;=\; \left\langle \mathrm{Tr}\big[(U^\mathsf{T} x^*)^\mathsf{T}(U^\mathsf{T} x^*)\big]\right\rangle = \mathrm{Tr}\Big[\langle x^* x^{*\mathsf{T}}\rangle U U^\mathsf{T}\Big]$$

$$=\; \mathrm{Tr}[\Sigma U U^\mathsf{T}] = \mathrm{Tr}[D U U^\mathsf{T}] = \sigma^2 \min(N,K) \qquad (8)$$

In summary, this means that the orthogonal basis $U$ computed from the training set spans all the variance in the training set but fails to do so on the test examples when $N < K$, i.e. for ill-posed data. The training set variance is $K/N\sigma^2$ on average per coordinate, compared to $\sigma^2$ for the test examples. So which of the two variances is "correct" ? From a modelling point of view, the variance from the test example tells us the true story, so the training set variance should be regarded as biased. This suggests that the training set singular values should be corrected for this bias, in the above example by re-estimating the training set projections using $\hat{Q} = \sqrt{N/K}Q$.

In the more general case we do not know $K$, and the true covariance may have an arbitrary eigen-spectrum. The GenSVD algorithm below is a more general algorithm for correcting for the training set bias.

## 2    The GenSVD Algorithm

The data matrix consists of $N$ statistically independent samples $X = \begin{bmatrix} x_1 \dots x_N \end{bmatrix}$ so $X$ is size $I \times N$, and each column of $X$ is assumed multivariate Gaussian, $x_j \sim \mathcal{N}(0,\Sigma)$ and is ill-posed with rank $\Sigma > N$.

With the SVD $X = U_0 \Lambda_0 V_0^\mathsf{T}$, we now make the approximation that $U_0$ contains an actual subset of the true eigen-vectors of $\Sigma$

$$\Sigma = \begin{bmatrix} U_0 \; U_\perp \end{bmatrix} \begin{bmatrix} \Lambda^2 & 0 \\ 0 & \Lambda_\perp^2 \end{bmatrix} \begin{bmatrix} U_0 \; U_\perp \end{bmatrix} \qquad (9)$$

where we have collected the remaining (unspanned by $X$) eigen-vectors and values in $U_\perp$ and $\Lambda_\perp^2$, satisfying $U_\perp^\mathsf{T} U_\perp = I$ and $U_0^\mathsf{T} U_\perp = 0$. The unknown 'true' eigen-values corresponding to the observed eigen-vectors are collected in $\Lambda = \mathrm{diag}\,(\lambda_1, \dots \lambda_N)$, which are the values we try to estimate in the following.

It should be noted that a direct estimation of $\Sigma$ using $\hat{\Sigma} = \frac{1}{N} X X^\mathsf{T}$ yields $\hat{\Sigma} = \frac{1}{N} U_0 \Lambda_0 V_0^\mathsf{T} V_0 \Lambda_0 U_0^\mathsf{T} = \frac{1}{N} U_0 \Lambda_0^2 U_0^\mathsf{T}$, i.e., the nonzero eigen-vectors and values of $\hat{\Sigma}$ is $U_0$ and $\Lambda_0$.

The distribution of test samples $x^*$ inside the space spanned by $U_0$ is

$$U_0^\mathsf{T} x \sim \mathcal{N}(0, U_0^\mathsf{T} \Sigma U_0) = \mathcal{N}(0, \Lambda^2) \qquad (10)$$

The problem is that $U_0$ and the examples $x_j$ are not independent, so $U_0^\top x_j$ is biased, e.g. the SVD estimate $\frac{1}{N}\Lambda_0^2$ of $\Lambda^2$ assigns all variance to lie within $U_0$.

The GenSVD algorithm bypasses this problem by, for each example, computing a basis on all other examples, estimating the variances in $\Lambda^2$ in a leave-one-out manner. Consider

$$z_j = U_0^\top B_{-j} B_{-j}^\top x_j \qquad (11)$$

where we introduce the notation $X_{-j}$ for the matrix of all examples except the $j$'th, and this matrix is decomposed as $X_{-j} = B_{-j}\Lambda_{-j}V_{-j}^\top$. The operation $B_{-j}B_{-j}^\top x_j$ projects the example onto the basis defined by the remaining examples, and back again, so it 'strips' off the part of signal space which is special for $x_j$ which could be signal which does not generalize across examples.

Since $B_{-j}$ and $x_j$ are independent $B_{-j}^\top x_j$ has the same distribution as the projection of a test example $x^*$, $B_{-j}^\top x^*$. Thus, $B_{-j}B_{-j}^\top x_j$ and $B_{-j}B_{-j}^\top x^*$ have the same distribution as well. Now, since span $B_{-j}$=span $X_{-j}$ and span $U_0$=span $[\,X_{-j}\,x_j\,]$ we have that span $B_{-j}\subseteq$span $U_0$ so we see that $z_j$ and $U_0^\top B_{-j}B_{-j}^\top x^*$ are identically distributed. This means that $z_j$ has the covariance $U_0^\top B_{-j}B_{-j}^\top \Sigma B_{-j}B_{-j}^\top U_0$ and using Eq. (9) and that $U_\perp^\top B_{-j} = 0$ (since $U_\perp^\top U_0 = 0$) we get

$$z_j \sim \mathcal{N}\big(0, U_0^\top B_{-j}B_{-j}^\top U_0 \Lambda U_0^\top B_{-j}B_{-j}^\top U_0\big) \qquad (12)$$

We note that this distribution is degenerate because the covariance is of rank $N-1$. For a sample $z_j$ from the above distribution we have that

$$U_0^\top B_{-j}B_{-j}^\top U_0 z_j = U_0^\top B_{-j}B_{-j}^\top U_0 U_0^\top B_{-j}B_{-j}^\top x_j = U_0^\top B_{-j}B_{-j}^\top x_j = z_j \qquad (13)$$

As a second approximation, assume that the observed $z_j$ are independent so that we can write the likelihood of $\Lambda$

$$
\begin{aligned}
-\log L(\Lambda^2) &\simeq \sum_j \log\left[(2\pi)^{N/2}\Big|(U_0^\top B_{-j})(B_{-j}^\top U_0)\Lambda^2(U_0^\top B_{-j})(B_{-j}^\top U_0)\Big|^{1/2}\right] \\
&\quad + \frac{1}{2}\sum_j z_j^\top (U_0^\top B_{-j})(B_{-j}^\top U_0)\Lambda^{-2}(U_0^\top B_{-j})(B_{-j}^\top U_0)z_j \\
&\simeq c + \frac{N}{2}\sum_j \log\lambda_j^2 + \frac{1}{2}\sum_j z_j^\top \Lambda^{-2} z_j \qquad (14)
\end{aligned}
$$

where we have used Eq. (13) and that the determinant[1] is approximated by $|\Lambda^2|$. This above expression is maximized when

$$\hat{\lambda}_i^2 = \frac{1}{N}\sum_j z_{ij}^2. \qquad (15)$$

The GenSVD of $X$ is then $X = U_0\hat{\Lambda}V^\top$, $\hat{\Lambda} = \mathrm{diag}(\hat{\lambda}_1,...,\hat{\lambda}_N)$.

In practice, using Eq. (11) directly to compute an SVD of the matrix $X_{-j}$ for each example is computationally demanding. It is possible to compute $z_j$ in a more efficient two-level procedure with the following algorithm:

Compute $U_0\Lambda_0 V_0^\top = \mathrm{svd}(X)$ and $Q_0 = [\,q_j\,] = \Lambda_0 V_0^\top$

foreach $j = 1...N$

      Compute $\boldsymbol{B}_{-j}\boldsymbol{\Lambda}_{-j}\boldsymbol{V}_{-j}^{\mathsf{T}} = \text{svd}(\boldsymbol{Q}_{-j})$

      $\boldsymbol{z}_j = \boldsymbol{B}_{-j}\boldsymbol{B}_{-j}^{\mathsf{T}}\boldsymbol{q}_j$

$\hat{\lambda}_i^2 = \frac{1}{N}\sum_j z_{ij}^2$

If the data has a mean value that we wish to remove prior to the SVD it is important that this is done within the GenSVD algorithm. Consider a centered matrix $\boldsymbol{X}_c = \boldsymbol{X} - \bar{\boldsymbol{X}}$ where $\bar{\boldsymbol{X}}$ contains the mean $\bar{x}$ replicated in all $N$ columns. The signal space in $\boldsymbol{X}_c$ is now corrupted because each centered example will contain a component of all examples, which means the 'stripping' of signal components not spanned by other examples no longer works: $\boldsymbol{B}_{-j}^{\mathsf{T}}\boldsymbol{x}_j$ is no longer distributed like $\boldsymbol{B}_{-j}^{\mathsf{T}}\boldsymbol{x}^*$. This suggests the alternative algorithm for data with removal of mean component:

Compute $\boldsymbol{U}_0\boldsymbol{\Lambda}_0\boldsymbol{V}_0^{\mathsf{T}} = \text{svd}(X)$ and $\boldsymbol{Q}_0 = \left[\,\boldsymbol{q}_j\,\right] = \boldsymbol{\Lambda}_0\boldsymbol{V}_0^{\mathsf{T}}$

foreach $j = 1...N$

      $\bar{\boldsymbol{q}}_{-j} = \frac{1}{N-1}\sum_{j'\neq j}\boldsymbol{q}_{j'}$

      Compute $\boldsymbol{B}_{-j}\boldsymbol{\Lambda}_{-j}\boldsymbol{V}_{-j}^{\mathsf{T}} = \text{svd}(\boldsymbol{Q}_{-j} - \bar{\boldsymbol{Q}}_{-j})$

      $\boldsymbol{z}_j = \boldsymbol{B}_{-j}\boldsymbol{B}_{-j}^{\mathsf{T}}(\boldsymbol{q}_j - \bar{\boldsymbol{q}}_{-j})$

$\hat{\lambda}_i^2 = \frac{1}{N-1}\sum_j z_{ij}^2$

Finally, note that it is possible to leave out more than one example at a time if the data is independent only in block, i.e. let $\boldsymbol{Q}_{-k}$ would be $\boldsymbol{Q}_0$ with the $k$'th block left out.

## Example With PET Scans

We compared the performance of GenSVD to conventional SVD on a functional $[^{15}\text{O}]$ water PET activation study of the human brain. The study consisted of 18 subjects, who were scanned four times while tracing a star-shaped maze with a joy-stick with visual feedback, in total 72 scans of dimension $\sim 25000$ spatial voxels. After the second scan, the visual feedback was mirrored, and the subject accomodated to and learned the new control environment during the last two scans. Scans were normalized by 1) dividing each scan by the average voxel value measured inside a brain mask and 2) for each scan subtracting the average scan for that subject thereby removing subject effects and 3) intra and inter-subject normalization and transformation using rigid body reorientation and affine linear transformations respectively. Voxels inside aforementioned brain mask were arranged in the data matrix with one scan per column.

Figure 1 shows the results of an SVD decomposition compared to GenSVD. Each marker represents one scan and the glyphs indicate scan number out of the four (circle-square-star-triangle). The ellipses indicate the mean and covariances of the projections in each scan number. The 32 scans from eight subjects were used as a training set and 40 scans from the remaining 10 subjects for testing. The training set projections are filled markers, test-set projections onto the basis defined by the training set are open markers (i.e. we plot the first two columns of $\boldsymbol{U}_0\boldsymbol{\Lambda}_0$ for SVD and of $\boldsymbol{U}_0\hat{\boldsymbol{\Lambda}}$ for GenSVD). We see that there is a clear difference in variance in the train- and test-examples, which is corrected quite well by GenSVD. The lower plot in Figure 1 shows the singular values for the PET data set. We see that GenSVD estimates are much closer to the actual test projection standard deviations than the SVD singular values.

# 3 Conclusion

We have demonstrated that projection of ill-posed data sets onto a basis defined by the same examples introduces a significant bias on the observed variance when comparing to projections of test examples onto the same basis. The GenSVD algorithm has been presented as a tool for correcting for this bias using a leave-one-out re-estimation scheme, and a computationally efficient implementation has been proposed.

We have demonstrated that the method works well on an ill-posed real-world data set, were the distribution of the GenSVD-corrected training test set projections matched the distribution of the observed test set projections far better than the uncorrected training examples. This allows a generalization performance increase of a subsequent statistical model, in the case of both supervised and unsupervised models.

## Acknowledgments

This work was supported partly by the Human Brain Project grant P20 MH57180, the Danish Research councils for the Natural and Technical Sciences through the Danish Computational Neural Network Center (CONNECT) and the Technology Center Through Highly Oriented Research (THOR).

## Footnotes

[1]Since $z_j$ is degenerate, we define the likelihood over the space where $z_j$ occur, i.e. the determinant in Eq. 14 should be read as 'the product of non-zero eigenvalues'.

## References

[Bishop, 1999] Bishop, C. (1999). Bayesian pca. In Kearns, M. S., Solla, S. A., and Cohn, D. A., editors, *Advances in Neural Information Processing Systems*, volume 11. The MIT Press.

[Hansen et al., 1999] Hansen, L., Larsen, J., Nielsen, F., Strother, S., Rostrup, E., Savoy, R., Lange, N., Sidtis, J., Svarer, C., and Paulson, O. (1999). Generalizable patterns in neuroimaging: How many principal components? *NeuroImage*, 9:534–544.

[Hansen and Larsen, 1996] Hansen, L. K. and Larsen, J. (1996). Unsupervised learning and generalization. In *Proceedings of IEEE International Conference on Neural Networks*, pages 25–30.

[Jackson, 1991] Jackson, J. E. (1991). *A User's Guide to Principal Components*. Wiley Series on Probability and Statistics, John Wiley and Sons.

[Lautrup et al., 1995] Lautrup, B., Hansen, L. K., Law, I., Mørch, N., Svarer, C., and Strother, S. (1995). Massive weight sharing: A cure for extremely ill-posed problems. In Hermann, H. J., Wolf, D. E., and Pöppel, E. P., editors, *Proceedings of Workshop on Supercomputing in Brain Research: From Tomography to Neural Networks: From tomography to neural networks, HLRZ, KFA Jülich, Germany*, pages 137–148. World Scientific.

[Roweis, 1998] Roweis, S. (1998). Em algorithms for pca and spca. In Jordan, M. I., Kearns, M. J., and Solla, S. A., editors, *Advances in Neural Information Processing Systems*, volume 10. The MIT Press.

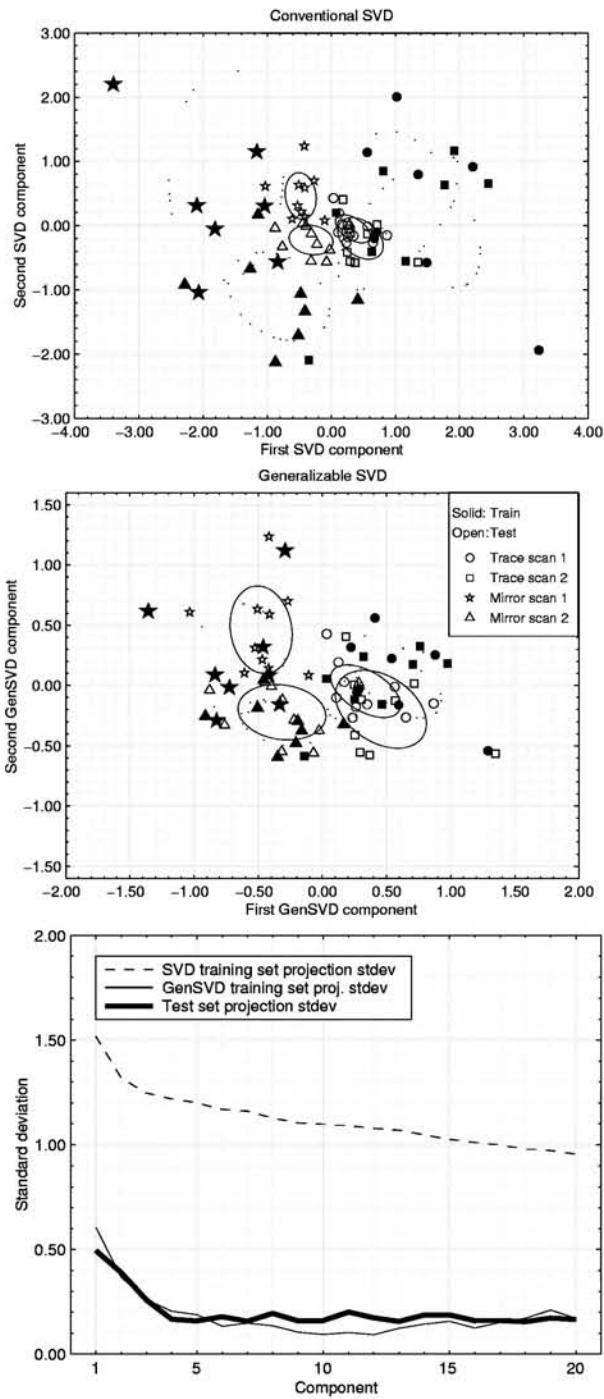

Figure 1: Projections of PET data in SVD and GenSVD. Each subject's four scans are indicated by: circle, square, star, triangle. Training set scans are marked with filled glyphs and test set with open glyphs. Solid and dotted Ellipses indicate test/train covariance per scan number. The third plot shows the standard deviations for the training and test set for SVD and GenSVD projections.